# Learning Halfspaces with the Zero-One Loss: Time-Accuracy Tradeoffs

**Aharon Birnbaum and Shai Shalev-Shwartz**
School of Computer Science and Engineering
The Hebrew University
Jerusalem, Israel

## Abstract

Given $\alpha, \epsilon$, we study the time complexity required to improperly learn a halfspace with misclassification error rate of at most $(1 + \alpha) L_\gamma^* + \epsilon$, where $L_\gamma^*$ is the optimal $\gamma$-margin error rate. For $\alpha = 1/\gamma$, polynomial time and sample complexity is achievable using the hinge-loss. For $\alpha = 0$, Shalev-Shwartz et al. [2011] showed that $\mathrm{poly}(1/\gamma)$ time is impossible, while learning is possible in time $\exp(\tilde{O}(1/\gamma))$. An immediate question, which this paper tackles, is what is achievable if $\alpha \in (0, 1/\gamma)$. We derive positive results interpolating between the polynomial time for $\alpha = 1/\gamma$ and the exponential time for $\alpha = 0$. In particular, we show that there are cases in which $\alpha = o(1/\gamma)$ but the problem is still solvable in polynomial time. Our results naturally extend to the adversarial online learning model and to the PAC learning with malicious noise model.

## 1 Introduction

Some of the most influential machine learning tools are based on the hypothesis class of halfspaces with margin. Examples include the Perceptron [Rosenblatt, 1958], Support Vector Machines [Vapnik, 1998], and AdaBoost [Freund and Schapire, 1997]. In this paper we study the computational complexity of learning halfspaces with margin.

A halfspace is a mapping $h(x) = \mathrm{sign}(\langle w, x \rangle)$, where $w, x \in \mathcal{X}$ are taken from the unit ball of an RKHS (e.g. $\mathbb{R}^n$), and $\langle w, x \rangle$ is their inner-product. Relying on the kernel trick, our sole assumption on $\mathcal{X}$ is that we are able to calculate efficiently the inner-product between any two instances (see for example Schölkopf and Smola [2002], Cristianini and Shawe-Taylor [2004]). Given an example $(x, y) \in \mathcal{X} \times \{\pm 1\}$ and a vector $w$, we say that $w$ errs on $(x, y)$ if $y\langle w, x \rangle \leq 0$ and we say that $w$ makes a $\gamma$-margin error on $(x, y)$ if $y\langle w, x \rangle \leq \gamma$.

The error rate of a predictor $h : \mathcal{X} \to \{\pm 1\}$ is defined as $L_{01}(h) = \mathbb{P}[h(x) \neq y]$, where the probability is over some unknown distribution over $\mathcal{X} \times \{\pm 1\}$. The $\gamma$-margin error rate of a predictor $x \mapsto \langle w, x \rangle$ is defined as $L_\gamma(w) = \mathbb{P}[y\langle w, x \rangle \leq \gamma]$. A learning algorithm $A$ receives an i.i.d. training set $S = (x_1, y_1), \ldots, (x_m, y_m)$ and its goal is to return a predictor, $A(S)$, whose error rate is small. We study the *runtime* required to learn a predictor such that with high probability over the choice of $S$, the error rate of the learnt predictor satisfies

$$L_{01}(A(S)) \leq (1 + \alpha) L_\gamma^* + \epsilon \quad \text{where} \quad L_\gamma^* = \min_{w:\|w\|=1} L_\gamma(w) . \qquad (1)$$

There are three parameters of interest: the margin parameter, $\gamma$, the multiplicative approximation factor parameter, $\alpha$, and the additive error parameter $\epsilon$.

From the statistical perspective (i.e., if we allow exponential runtime), Equation (1) is achievable with $\alpha = 0$ by letting $A$ be the algorithm which minimizes the number of margin errors over the

training set subject to a norm constraint on $w$. The sample complexity of $A$ is $m = \tilde{\Omega}(\frac{1}{\gamma^2\epsilon^2})$. See for example Cristianini and Shawe-Taylor [2004].

If the data is separable with margin (that is, $L_\gamma^* = 0$), then the aforementioned $A$ can be implemented in time $\mathrm{poly}(1/\gamma, 1/\epsilon)$. However, the problem is much harder in the agnostic case, namely, when $L_\gamma^* > 0$ and the distribution over examples can be arbitrary.

Ben-David and Simon [2000] showed that, no proper learning algorithm can satisfy Equation (1) with $\alpha = 0$ while running in time polynomial in both $1/\gamma$ and $1/\epsilon$. By "proper" we mean an algorithm which returns a halfspace predictor. Shalev-Shwartz et al. [2011] extended this results to improper learning—that is, when $A(S)$ should satisfy Equation (1) but is not required to be a halfspace. They also derived an algorithm that satisfies Equation (1) and runs in time $\exp\left(C\frac{1}{\gamma}\log(\frac{1}{\gamma\epsilon})\right)$, where $C$ is a constant.

Most algorithms that are being used in practice minimize a convex surrogate loss. That is, instead of minimizing the number of mistakes on the training set, the algorithms minimize $\hat{L}(w) = \frac{1}{m}\sum_{i=1}^m \ell(y_i\langle w, x_i\rangle)$, where $\ell : \mathbb{R} \to \mathbb{R}$ is a convex function that upper bounds the $0-1$ loss. For example, the Support Vector Machine (SVM) algorithm relies on the hinge loss. The advantage of surrogate convex losses is that minimizing them can be performed in time $\mathrm{poly}(1/\gamma, 1/\epsilon)$. It is easy to verify that minimizing $\hat{L}(w)$ with respect to the hinge loss yields a predictor that satisfies Equation (1) with $\alpha = \frac{1}{\gamma}$. Furthermore, Long and Servedio [2011], Ben-David et al. [2012] have shown that any convex surrogate loss cannot guarantee Equation (1) if $\alpha < \frac{1}{2}\left(\frac{1}{\gamma} - 1\right)$.

Despite the centrality of this problem, not much is known on the runtime required to guarantee Equation (1) with other values of $\alpha$. In particular, a natural question is how the runtime changes when enlarging $\alpha$ from 0 to $\frac{1}{\gamma}$. Does it change gradually or perhaps there is a phase transition?

Our main contribution is an upper bound on the required runtime as a function of $\alpha$. For any $\alpha$ between[1] 5 and $\frac{1}{\gamma}$, let $\tau = \frac{1}{\gamma\alpha}$. We show that the runtime required to guarantee Equation (1) is at most $\exp(C\tau\min\{\tau, \log(1/\gamma)\})$, where $C$ is a universal constant (we ignore additional factors which are polynomial in $1/\gamma, 1/\epsilon$—see a precise statement with the exact constants in Theorem 1). That is, when we enlarge $\alpha$, the runtime decreases gradually from being exponential to being polynomial.

Furthermore, we show that the algorithm which yields the aforementioned bound is a vanilla SVM with a specific kernel. We also show how one can design specific kernels that will fit well certain values of $\alpha$ while minimizing our upper bound on the sample and time complexity.

In Section 4 we extend our results to the more challenging learning settings of adversarial online learning and PAC learning with malicious noise. For both cases, we obtain similar upper bounds on the runtime as a function of $\alpha$. The technique we use in the malicious noise case may be of independent interest.

An interesting special case is when $\alpha = \frac{1}{\gamma\sqrt{\log(1/\gamma)}}$. In this case, $\tau = \sqrt{\log(1/\gamma)}$ and hence the runtime is still polynomial in $1/\gamma$. This recovers a recent result of Long and Servedio [2011]. Their technique is based on a smooth boosting algorithm applied on top of a weak learner which constructs random halfspaces and takes their majority vote. Furthermore, Long and Servedio emphasize that their algorithm is not based on convex optimization. They show that no convex surrogate can obtain $\alpha = o(1/\gamma)$. As mentioned before, our technique is rather different as we do rely on the hinge loss as a surrogate convex loss. There is no contradiction to Long and Servedio since we apply the convex loss in the feature space induced by our kernel function. The negative result of Long and Servedio holds only if the convex surrogate is applied on the original space.

## 1.1 Additional related work

The problem of learning kernel-based halfspaces has been extensively studied before in the framework of SVM [Vapnik, 1998, Cristianini and Shawe-Taylor, 2004, Schölkopf and Smola, 2002] and the Perceptron [Freund and Schapire, 1999]. Most algorithms replace the 0-1 error function with a convex surrogate. As mentioned previously, Ben-David et al. [2012] have shown that this approach leads to approximation factor of at least $\frac{1}{2}\left(\frac{1}{\gamma} - 1\right)$.

There has been several works attempting to obtain efficient algorithm for the case $\alpha = 0$ under certain distributional assumptions. For example, Kalai et al. [2005], Blais et al. [2008] have shown that if the marginal data distribution over $\mathcal{X}$ is a product distribution, then it is possible to satisfy Equation (1) with $\alpha = \gamma = 0$, in time $\mathrm{poly}(n^{1/\epsilon^4})$. Klivans et al. [2009] derived similar results for the case of malicious noise. Another distributional assumption is on the conditional probability of the label given the instance. For example, Kalai and Sastry [2009] solves the problem in polynomial time if there exists a vector $w$ and a monotonically non-increasing function $\phi$ such that $\mathbb{P}(Y = 1|X = x) = \phi(\langle w, x \rangle)$.

Zhang [2004], Bartlett et al. [2006] also studied the relationship between surrogate convex loss functions and the 0-1 loss function. They introduce the notion of well calibrated loss functions, meaning that the excess risk of a predictor $h$ (over the Bayes optimal) with respect to the 0-1 loss can be bounded using the excess risk of the predictor with respect to the surrogate loss. It follows that if the latter is close to zero than the former is also close to zero. However, as Ben-David et al. [2012] show in detail, without making additional distributional assumptions the fact that a loss function is well calibrated does not yield finite-sample or finite-time bounds.

In terms of techniques, our Theorem 1 can be seen as a generalization of the positive result given in Shalev-Shwartz et al. [2011]. While Shalev-Shwartz et al. only studied the case $\alpha = 0$, we are interested in understanding the whole curve of runtime as a function of $\alpha$. Similar to the analysis of Shalev-Shwartz et al., we approximate the sigmoidal and erf transfer functions using polynomials. However, we need to break symmetry in the definition of the exact transfer function to approximate. The main technical observation is that the Lipschitz constant of the transfer functions we approximate does not depend on $\alpha$, and is roughly $1/\gamma$ no matter what $\alpha$ is. Instead, the change of the transfer function when $\alpha$ is increasing is in higher order derivatives.

To the best of our knowledge, the only middle point on the curve that has been studied before is the case $\alpha = \frac{1}{\gamma\sqrt{\log(1/\gamma)}}$, which was analyzed in Long and Servedio [2011]. Our work shows an upper bound on the entire curve. Besides that, we also provide a recipe for constructing better kernels for specific values of $\alpha$.

## 2 Main Results

Our main result is an upper bound on the time and sample complexity for all values of $\alpha$ between $5$ and $1/\gamma$. The bounds we derive hold for a norm-constraint form of SVM with a specific kernel, which we recall now. Given a training set $S = (x_1, y_1), \ldots, (x_m, y_m)$, and a feature mapping $\psi : \mathcal{X} \to \mathcal{X}'$, where $\mathcal{X}'$ is the unit ball of some Hilbert space, consider the following learning rule:

$$\operatorname*{argmin}_{v:\|v\|^2 \leq B} \sum_{i=1}^{m} \max\{0, 1 - y_i \langle v, \psi(x_i) \rangle\} . \tag{2}$$

Using the well known kernel-trick, if $K(x, x')$ implements the inner product $\langle \psi(x), \psi(x') \rangle$, and $G$ is an $m \times m$ matrix with $G_{i,j} = K(x_i, x_j)$, then we can write a solution of Equation (2) as $v = \sum_i a_i \psi(x_i)$ where the vector $a \in \mathbb{R}^m$ is a solution of

$$\operatorname*{argmin}_{a:a^T G a \leq B} \sum_{i=1}^{m} \max\{0, 1 - y_i (Ga)_i\} . \tag{3}$$

The above is a convex optimization problem in $m$ variables and can be solved in time $\mathrm{poly}(m)$. Given a solution $a \in \mathbb{R}^m$, we define a classifier $h_a : X \to \{\pm 1\}$ to be

$$h_a(x) = \mathrm{sign}\left(\sum_{i=1}^{m} a_i K(x_i, x)\right) . \tag{4}$$

The upper bounds we derive hold for the above kernel-based SVM with the kernel function

$$K(x, x') = \frac{1}{1 - \frac{1}{2}\langle x, x'\rangle} \ . \tag{5}$$

We are now ready to state our main theorem.

**Theorem 1** *For any $\gamma \in (0, 1/2)$ and $\alpha \geq 5$, let $\tau = \frac{1}{\gamma\alpha}$ and let*

$$B = \min\left\{ 4\alpha^2\left(96\tau^2 + e^{18\tau\log(8\tau\alpha^2)+5}\right)\ ,\ \frac{1}{\gamma^2}\left(0.06\,e^{4\tau^2} + 3\right)\right\}$$

$$= \text{poly}(1/\gamma)\cdot e^{\min\left\{18\tau\log(8\tau\alpha^2)\,,\,4\tau^2\right\}} \ .$$

*Fix $\epsilon, \delta \in (0, 1/2)$ and let $m$ be a training set size that satisfies*

$$m \ \geq\ \frac{16}{\epsilon^2}\max\{2B, (1+\alpha)^2\log(2/\delta)\} \ .$$

*Let $A$ be the algorithm which solves Equation (3) with the kernel function given in Equation (5), and returns the predictor defined in Equation (4). Then, for any distribution, with probability of at least $1 - \delta$, the algorithm $A$ satisfies Equation (1).*

The proof of the theorem is given in the next section. As a direct corollary we obtain that there is an efficient algorithm that achieves an approximation factor of $\alpha = o(1/\gamma)$:

**Corollary 2** *For any $\epsilon, \delta, \gamma \in (0, 1)$, let $\alpha = \frac{1/\gamma}{\sqrt{\log(1/\gamma)}}$ and let $B = \frac{0.06}{\gamma^6} + \frac{3}{\gamma^2}$. Then, with $m, A$ being as defined in Theorem 1, the algorithm $A$ satisfies Equation (1).*

As another corollary of Theorem 1 we obtain that for any constant $c \in (0, 1)$, it is possible to satisfy Equation (1) with $\alpha = c/\gamma$ in polynomial time. However, the dependence of the runtime on the constant $c$ is $e^{4/c^2}$. For example, for $c = 1/2$ we obtain the multiplicative factor $e^{16} \approx 8,800,000$. Our next contribution is to show that a more careful design of the kernel function can yield better bounds.

**Theorem 3** *For any $\gamma, \alpha$, let $p$ be a polynomial of the form $p(z) = \sum_{j=1}^{d}\beta_j z^{2j-1}$ (namely, $p$ is odd) that satisfies*

$$\max_{z\in[-1,1]}|p(z)| \leq \alpha \quad and \quad \min_{z:|z|\geq\gamma}|p(z)| \geq 1 \ .$$

*Let $m$ be a training set size that satisfies*

$$m \ \geq\ \frac{16}{\epsilon^2}\ \max\{\|\beta\|_1^2, 2\log(4/\delta), (1+\alpha)^2\log(2/\delta)\}$$

*Let $A$ be the algorithm which solves Equation (3) with the following kernel function*

$$K(x, x') = \sum_{j=1}^{d}|\beta_j|(\langle x, x'\rangle)^{2j-1} \ ,$$

*and returns the predictor defined in Equation (4). Then, for any distribution, with probability of at least $1 - \delta$, the algorithm $A$ satisfies Equation (1).*

The above theorem provides us with a recipe for constructing good kernel functions: Given $\gamma$ and $\alpha$, find a vector $\beta$ with minimal $\ell_1$ norm such that the polynomial $p(z) = \sum_{j=1}^{d}\beta_j z^{2j-1}$ satisfies the conditions given in Theorem 3. For a fixed degree $d$, this can be written as the following optimization problem:

$$\min_{\beta\in\mathbb{R}^d} \|\beta\|_1 \quad \text{s.t.} \quad \forall x \in [0, 1],\ p(z) \leq \alpha \quad \wedge \quad \forall z \in [\gamma, 1],\ p(z) \geq 1 \ . \tag{6}$$

Note that for any $x$, the expression $p(x)$ is a linear function of $\beta$. Therefore, the above problem is a linear program with an infinite number of constraints. Nevertheless, it can be solved efficiently using the Ellipsoid algorithm. Indeed, for any $\beta$, we can find the extreme points of the polynomial

it defines, and then determine whether $\beta$ satisfies all the constraints or, if it doesn't, we can find a violated constraint.

To demonstrate how Theorem 3 can yield a better guarantee (in terms of the constants), we solved Equation (6) for the simple case of $d = 2$. For this simple case, we can provide an analytic solution to Equation (6), and based on this solution we obtain the following lemma whose proof is provided in the appendix.

**Lemma 4** *Given $\gamma < 2/3$, consider the polynomial $p(z) = \beta_1 z + \beta_2 z^3$, where*
$$\beta_1 = \frac{1}{\gamma} + \frac{\gamma}{1+\gamma} \quad , \quad \beta_2 = -\frac{1}{\gamma(1+\gamma)} \ .$$
*Then, $p$ satisfies the conditions of Theorem 3 with*
$$\alpha = \frac{2}{3\sqrt{3\gamma}} + \frac{2}{\sqrt{3}} \leq 0.385 \cdot \frac{1}{\gamma} + 1.155 \ .$$

*Furthermore, $\|\beta\|_1 \leq \frac{2}{\gamma} + 1$.*

It is interesting to compare the guarantee given in the above lemma to the guarantee of using the vanilla hinge-loss. For both cases the sample complexity is order of $\frac{1}{\gamma^2\epsilon^2}$. For the vanilla hinge-loss we obtain the approximation factor $\frac{1}{\gamma}$ while for the kernel given in Lemma 4 we obtain the approximation factor of $\alpha \leq 0.385 \cdot \frac{1}{\gamma} + 1.155$. Recall that Ben-David et al. [2012] have shown that without utilizing kernels, no convex surrogate loss can guarantee an approximation factor smaller than $\alpha < \frac{1}{2}(\frac{1}{\gamma} - 1)$. The above discussion shows that applying the hinge-loss with a kernel function can break this barrier without a significant increase in runtime[2] or sample complexity.

## 3 Proofs

Given a scalar loss function $\ell : \mathbb{R} \to \mathbb{R}$, and a vector $w$, we denote by $L(w) = \mathbb{E}_{(x,y)\sim\mathcal{D}}[\ell(y\langle w, x\rangle)]$ the expected loss value of the predictions of $w$ with respect to a distribution $\mathcal{D}$ over $\mathcal{X} \times \{\pm 1\}$. Given a training set $S = (x_1, y_1), \ldots, (x_m, y_m)$, we denote by $\hat{L}(w) = \frac{1}{m}\sum_{i=1}^{m} \ell(y_i\langle w, x_i\rangle)$ the empirical loss of $w$. We slightly overload our notation and also use $L(w)$ to denote $\mathbb{E}_{(x,y)\sim\mathcal{D}}[\ell(y\langle w, \psi(x)\rangle)]$, when $w$ is an element of an RKHS corresponding to the mapping $\psi$. We define $\hat{L}(w)$ analogously.

We will make extensive use of the following loss functions: the zero-one loss, $\ell_{01}(z) = 1[z \leq 0]$, the $\gamma$-zero-one loss, $\ell_{\gamma}(z) = 1[z \leq \gamma]$, the hinge-loss, $\ell_h(z) = [1-z]_+ = \max\{0, 1-z\}$, and the ramp-loss, $\ell_{\text{ramp}}(z) = \min\{1, \ell_h(z)\}$. We will use $L_{01}(w), L_{\gamma}(w), L_h(w)$, and $L_{\text{ramp}}(w)$ to denote the expectations with respect to the different loss functions. Similarly $\hat{L}_{01}(w), \hat{L}_{\gamma}(w), \hat{L}_h(w)$, and $\hat{L}_{\text{ramp}}(w)$ are the empirical losses of $w$ with respect to the different loss functions.

Recall that we output a vector $v$ that solves Equation (3). This vector is in the RKHS corresponding to the kernel given in Equation (5). Let $B_x = \max_{x\in\mathcal{X}} K(x, x) \leq 2$. Since the ramp-loss upper bounds the zero-one loss we have that $L_{01}(v) \leq L_{\text{ramp}}(v)$. The advantage of using the ramp loss is that it is both a Lipschitz function and it is bounded by 1. Hence, standard Rademacher generalization analysis (e.g. Bartlett and Mendelson [2002], Bousquet [2002]) yields that with probability of at least $1 - \delta/2$ over the choice of $S$ we have:

$$L_{\text{ramp}}(v) \leq \hat{L}_{\text{ramp}}(v) + \underbrace{2\sqrt{\frac{B_x B}{m}} + \sqrt{\frac{2\ln(4/\delta)}{m}}}_{=\epsilon_1} \ . \tag{7}$$

Since the ramp loss is upper bounded by the hinge-loss, we have shown the following inequalities,

$$L_{01}(v) \leq L_{\text{ramp}}(v) \leq \hat{L}_{\text{ramp}}(v) + \epsilon_1 \leq \hat{L}_h(v) + \epsilon_1 \ . \tag{8}$$

Next, we rely on the following claim adapted from [Shalev-Shwartz et al., 2011, Lemma 2.4]:

**Claim 5** *Let $p(z) = \sum_{j=0}^{\infty} \beta_j z^j$ be any polynomial that satisfies $\sum_{j=0}^{\infty} \beta_j^2 2^j \leq B$, and let $w$ be any vector in $\mathcal{X}$. Then, there exists $v_w$ in the RKHS defined by the kernel given in Equation (5), such that $\|v_w\|^2 \leq B$ and for all $x \in \mathcal{X}$, $\langle v_w, \psi(x) \rangle = p(\langle w, x \rangle)$.*

For any polynomial $p$, let $\ell_p(z) = \ell_h(p(z))$, and let $\hat{L}_p$ be defined analogously. If $p$ is an odd polynomial, we have that $\ell_p(y\langle w, x \rangle) = [1 - yp(\langle w, x \rangle)]_+$. By the definition of $v$ as minimizing $\hat{L}_h(v)$ over $\|v\|^2 \leq B$, it follows from the above claim that for any odd $p$ that satisfies $\sum_{j=0}^{\infty} \beta_j^2 2^j \leq B$ and for any $w^* \in X$, we have that

$$\hat{L}_h(v) \leq \hat{L}_h(v_{w^*}) = \hat{L}_p(w^*) .$$

Next, it is straightforward to verify that if $p$ is an odd polynomial that satisfies:

$$\max_{z \in [-1,1]} |p(z)| \leq \alpha \quad \text{and} \quad \min_{z \in [\gamma,1]} p(z) \geq 1 \tag{9}$$

then, $\ell_p(z) \leq (1 + \alpha)\ell_\gamma(z)$ for all $z \in [-1, 1]$. For such polynomials, we have that $\hat{L}_p(w^*) \leq (1 + \alpha)\hat{L}_\gamma(w^*)$. Finally, by Hoeffding's inequality, for any fixed $w^*$, if $m > \frac{\log(2/\delta)}{\epsilon_2^2}$, then with probability of at least $1 - \delta/2$ over the choice of $S$ we have that

$$\hat{L}_\gamma(w^*) \leq L_\gamma(w^*) + \epsilon_2 .$$

So, overall, we have obtained that with probability of at least $1 - \delta$,

$$L_{01}(v) \leq (1 + \alpha) L_\gamma(w^*) + (1 + \alpha)\epsilon_2 + \epsilon_1 .$$

Choosing $m$ large enough so that $(1 + \alpha)\epsilon_2 + \epsilon_1 \leq \epsilon$, we obtain:

**Corollary 6** *Fix $\gamma, \epsilon, \delta \in (0, 1)$ and $\alpha > 0$. Let $p$ be an odd polynomial such that $\sum_j \beta_j^2 2^j \leq B$ and such that Equation (9) holds. Let $m$ be a training set size that satisfies:*

$$m \geq \frac{16}{\epsilon^2} \cdot \max\{2B, 2\log(4/\delta), (1 + \alpha)^2 \log(2/\delta)\} .$$

*Then, with probability of at least $1 - \delta$, the solution of Equation (3) satisfies, $L_{01}(v) \leq (1+\alpha)L_\gamma^* + \epsilon$.*

The proof of Theorem 1 follows immediately from the above corollary together with the following two lemmas, whose proofs are provided in the appendix.

**Lemma 7** *For any $\gamma > 0$ and $\alpha > 2$, let $\tau = \frac{1}{\alpha\gamma}$ and let $B = \frac{1}{\gamma^2}\left(0.06\, e^{4\tau^2} + 3\right)$. Then, there exists a polynomial that satisfies the conditions of Corollary 6 with the parameters $\alpha, \gamma, B$.*

**Lemma 8** *For any $\gamma \in (0, 1/2)$ and $\alpha \in [5, \frac{1}{\gamma}]$, let $\tau = \frac{1}{\alpha\gamma}$ and let $B = 4\alpha^2\left(96\tau^2 + \exp\left(18\tau\log\left(8\tau\alpha^2\right) + 5\right)\right)$. Then, there exists a polynomial that satisfies the conditions of Corollary 6 with the parameters $\alpha, \gamma, B$.*

### 3.1 Proof of Theorem 3

The proof is similar to the proof of Theorem 1 except that we replace Claim 5 with the following:

**Lemma 9** *Let $p(z) = \sum_{j=1}^{d} \beta_j z^{2j-1}$ be any polynomial, and let $w$ be any vector in $\mathcal{X}$. Then, there exists $v_w$ in the RKHS defined by the kernel given in Theorem 3, such that $\|v_w\|^2 \leq \|\beta\|_1$ and for all $x \in \mathcal{X}$, $\langle v_w, \psi(x) \rangle = p(\langle w, x \rangle)$.*

**Proof** We start with an explicit definition of the mapping $\psi(x)$ corresponding to the kernel in the theorem. The coordinates of $\psi(x)$ are indexed by tuples $(k_1, \ldots, k_j) \in [n]^j$ for $j = 1, 3, \ldots, 2d-1$. Coordinate $(k_1, \ldots, k_j)$ equals to $\sqrt{|\beta_j|}x_{k_1}x_{k_2}\ldots x_{k_j}$. Next, for any $w \in \mathcal{X}$, we define explicitly the vector $v_w$ for which $\langle v_w, \psi(x) \rangle = p(\langle w, x \rangle)$. Coordinate $(k_1, \ldots k_j)$ of $v_w$ equals to $\text{sign}(\beta_j)\sqrt{|\beta_j|}w_{k_1}w_{k_2}\ldots w_{k_j}$. It is easy to verify that indeed $\|v_w\|^2 \leq \|\beta\|_1$ and for all $x \in \mathcal{X}$, $\langle v_w, \psi(x) \rangle = p(\langle w, x \rangle)$. ∎

Since for any $x \in \mathcal{X}$ we also have that $K(x, x) \leq \|\beta\|_1$, the proof of Theorem 3 follows using the same arguments as in the proof of Theorem 1.

# 4 Extension to other learning models

In this section we briefly describe how our results can be extended to adversarial online learning and to PAC learning with malicious noise. We start with the online learning model.

## 4.1 Online learning

Online learning is performed in a sequence of consecutive rounds, where at round $t$ the learner is given an instance, $x_t \in \mathcal{X}$, and is required to predict its label. After predicting $\hat{y}_t$, the target label, $y_t$, is revealed. The goal of the learner is to make as few prediction mistakes as possible. See for example Cesa-Bianchi and Lugosi [2006].

A classic online classification algorithm is the Perceptron [Rosenblatt, 1958]. The Perceptron maintains a vector $w_t$ and predicts according to $\hat{y}_t = \text{sign}(\langle w_t, x_t \rangle)$. Initially, $w_1 = 0$, and at round $t$ the Perceptron updates the vector using the rule $w_{t+1} = w_t + 1[\hat{y}_t \neq y_t] y_t x_t$. Freund and Schapire [1999] observed that the Perceptron can also be implemented efficiently in an RKHS using a kernel function.

Agmon [1954] and others have shown that if there exists $w^*$ such that for all $t$, $y_t \langle w^*, x_t \rangle \geq 1$ and $\|x_t\|^2 \leq B_x$, then the Perceptron will make at most $\|w^*\|^2 B_x$ prediction mistakes. This bound holds without making any additional distributional assumptions on the sequence of examples.

This mistake bound has been generalized to the noisy case (see for example Gentile [2003]) as follows. Given a sequence $(x_1, y_1), \ldots, (x_m, y_m)$, and a vector $w^*$, let $L_h(w^*) = \frac{1}{m} \sum_{t=1}^{m} \ell_h(y_t \langle w^*, x_t \rangle)$, where $\ell_h$ is the hinge-loss. Then, the average number of prediction mistakes the Perceptron will make on this sequence is at most

$$\frac{1}{m} \sum_{t=1}^{m} 1[\hat{y}_t \neq y_t] \leq L_h(w^*) + \sqrt{\frac{B_x \|w^*\|^2 L_h(w^*)}{m}} + \frac{B_x \|w^*\|^2}{m} . \tag{10}$$

Let $L_\gamma(w^*) = \frac{1}{m} \sum_{t=1}^{m} 1(y_t \langle w^*, x_t \rangle \leq \gamma)$. Trivially, Equation (10) can yield a bound whose leading term is $\left(1 + \frac{1}{\gamma}\right) L_\gamma(w^*)$ (namely, it corresponds to $\alpha = \frac{1}{\gamma}$). On the other hand, Ben-David et al. [2009] have derived a mistake bound whose leading term depends on $L_\gamma(w^*)$ (namely, it corresponds to $\alpha = 0$), but the runtime of the algorithm is at least $m^{1/\gamma^2}$. The main result of this section is to derive a mistake bound for the Perceptron based on all values of $\alpha$ between 5 and $1/\gamma$.

**Theorem 10** *For any $\gamma \in (0, 1/2)$ and $\alpha \geq 5$, let $\tau = \frac{1}{\gamma \alpha}$ and let $B_{\alpha, \gamma}$ be the value of $B$ as defined in Theorem 1. Then, for any sequence $(x_1, y_1), \ldots, (x_m, y_m)$, if the Perceptron is run on this sequence using the kernel function given in Equation (5), the average number of prediction mistakes it will make is at most:*

$$\min_{\gamma \in (0, 1/2), \alpha \geq 5, w^* \in X} (1 + \alpha) L_\gamma(w^*) + \sqrt{\frac{2 B_{\alpha, \gamma} (1 + \alpha) L_\gamma(w^*)}{m}} + \frac{2 B_{\alpha, \gamma}}{m}$$

**Proof** [sketch] Equation (10) holds if we implement the Perceptron using the kernel function given in Equation (5), for which $B_x = 2$. Furthermore, similarly to the proof of Theorem 1, for any polynomial $p$ that satisfies the conditions of Corollary 6 we have that there exists $v^*$ in the RKHS corresponding to the kernel, with $\|v^*\|^2 \leq B$ and with $L_h(v^*) \leq (1 + \alpha) L_\gamma(w^*)$. The theorem follows. ∎

## 4.2 PAC learning with malicious noise

In this model, introduced by Valiant [1985] and specified to the case of halfspaces with margin by Servedio [2003], Long and Servedio [2011], there is an unknown distribution over instances in $\mathcal{X}$ and there is an unknown target vector $w^* \in \mathcal{X}$ such that $|\langle w^*, x \rangle| \geq \gamma$ with probability 1. The learner has an access to an example oracle. At each query to the oracle, with probability of $1 - \eta$ it samples a random example $x \in \mathcal{X}$ according to the unknown distribution over $\mathcal{X}$, and

returns $(x, \text{sign}(\langle w^*, x \rangle))$. However, with probability $\eta$, the oracle returns an arbitrary element of $\mathcal{X} \times \{\pm 1\}$. The goal of the learner is to output a predictor that has $L_{01}(h) \leq \epsilon$, with respect to the "clean" distribution.

Auer and Cesa-Bianchi [1998] described a general conversion from online learning to the malicious noise setting. Servedio [2003] used this conversion to derive a bound based on the Perceptron's mistake bound. In our case, we cannot rely on the conversion of Auer and Cesa-Bianchi [1998] since it requires a proper learner, while the online learner described in the previous section is not proper.

Instead, we propose the following simple algorithm. First, sample $m$ examples. Then, solve kernel SVM on the resulting noisy training set.

**Theorem 11** *Let $\gamma \in (0, 1/4)$, $\delta \in (0, 1/2)$, and $\alpha > 5$. Let $B$ be as defined in Theorem 1. Let $m$ be a training set size that satisfies: $m \geq \frac{64}{\epsilon^2} \cdot \max\left\{ 2B \,,\, (2 + \alpha)^2 \log(1/\delta) \right\}$. Then, with probability of at least $1 - 2\delta$, the output of kernel-SVM on the noisy training set, denoted $h$, satisfies $L_{01}(h) \leq (2 + \alpha)\eta + \epsilon/2$. It follows that if $\eta \leq \frac{\epsilon}{2(2+\alpha)}$ then $L_{01}(h) \leq \epsilon$.*

**Proof** Let $\bar{S}$ be a training set in which we replace the noisy examples with clean iid examples. Let $\bar{L}$ denotes the empirical loss over $\bar{S}$ and $\hat{L}$ denotes the empirical loss over $S$. As in the proof of Theorem 1, we have that w.p. of at least $1 - \delta$, for any $v$ in the RKHS corresponding to the kernel that satisfies $\|v\|^2 \leq B$ we have that:

$$L_{01}(v) \leq \bar{L}_{\text{ramp}}(v) + 3\epsilon/8 \,, \tag{11}$$

by our assumption on $m$. Let $\hat{\eta}$ be the fraction of noisy examples in $S$. Note that $\bar{S}$ and $S$ differ in at most $m\hat{\eta}$ elements. Therefore, for any $v$,

$$\bar{L}_{\text{ramp}}(v) \leq \hat{L}_{\text{ramp}}(v) + \hat{\eta} \,. \tag{12}$$

Now, let $v$ be the minimizer of $\hat{L}_h$, let $w^*$ be the target vector in the original space (i.e., the one which achieves correct predictions with margin $\gamma$ on clean examples), and let $v_{w^*}$ be its corresponding element in the RKHS (see Claim 5). We have

$$\hat{L}_{\text{ramp}}(v) \leq \hat{L}_h(v) \leq \hat{L}_h(v_{w^*}) = \hat{L}_p(w^*) \leq (1 + \alpha)\hat{L}_\gamma(w^*) \leq (1 + \alpha)\hat{\eta} \,. \tag{13}$$

In the above, the first inequality is since the ramp loss is upper bounded by the hinge loss, the second inequality is by the definition of $v$, the third equality is by Claim 5, the fourth inequality is by the properties of $p$, and the last inequality follows from the definition of $\hat{\eta}$. Combining the above yields,

$$L_{01}(v) \leq (2 + \alpha)\hat{\eta} + 3\epsilon/8 \,.$$

Finally, using Hoefding's inequality, we know that for the definition of $m$, with probability of at least $1 - \delta$ we have that $\hat{\eta} \leq \eta + \frac{\epsilon}{8(2+\alpha)}$. Applying the union bound and combining the above we conclude that with probability of at least $1 - 2\delta$, $L_{01}(v) \leq (2 + \alpha)\eta + \epsilon/2$. ∎

## 5  Summary and Open Problems

We have derived upper bounds on the time and sample complexities as a function of the approximation factor. We further provided a recipe for designing kernel functions with a small time and sample complexity for any given value of approximation factor and margin. Our results are applicable to agnostic PAC Learning, online learning, and PAC learning with malicious noise.

An immediate open question is whether our results can be improved. If not, can computationally hardness results be formally established. Another open question is whether the upper bounds we have derived for an improper learner can be also derived for a proper learner.

**Acknowledgements:** This work is supported by the Israeli Science Foundation grant number 598-10 and by the German-Israeli Foundation grant number 2254-2010. Shai Shalev-Shwartz is incumbent of the John S. Cohen Senior Lectureship in Computer Science.

## Footnotes

[1]We did not analyze the case $\alpha < 5$ because the runtime is already exponential in $1/\gamma$ even when $\alpha = 5$. Note, however, that our bound for $\alpha = 5$ is slightly better than the bound of Shalev-Shwartz et al. [2011] for $\alpha = 0$ because our bound does not involve the parameter $\epsilon$ in the exponent while their bound depends on $\exp(1/\gamma\log(1/(\epsilon\gamma)))$.

[2]It should be noted that solving SVM with kernels takes more time than solving a linear SVM. Hence, if the original instance space is a low dimensional Euclidean space we loose polynomially in the time complexity. However, when the original instance space is also an RKHS, and our kernel is composed on top of the original kernel, the increase in the time complexity is not significant.

# References

S. Agmon. The relaxation method for linear inequalities. *Canadian Journal of Mathematics*, 6(3):382–392, 1954.

P. Auer and N. Cesa-Bianchi. On-line learning with malicious noise and the closure algorithm. *Annals of mathematics and artificial intelligence*, 23(1):83–99, 1998.

P. L. Bartlett and S. Mendelson. Rademacher and Gaussian complexities: Risk bounds and structural results. *Journal of Machine Learning Research*, 3:463–482, 2002.

P. L. Bartlett, M. I. Jordan, and J. D. McAuliffe. Convexity, classification, and risk bounds. *Journal of the American Statistical Association*, 101:138–156, 2006.

S. Ben-David and H. Simon. Efficient learning of linear perceptrons. In *NIPS*, 2000.

S. Ben-David, D. Pal, , and S. Shalev-Shwartz. Agnostic online learning. In *COLT*, 2009.

S. Ben-David, D. Loker, N. Srebro, and K. Sridharan. Minimizing the misclassification error rate using a surrogate convex loss. In *ICML*, 2012.

E. Blais, R. O'Donnell, and K Wimmer. Polynomial regression under arbitrary product distributions. In *COLT*, 2008.

O. Bousquet. *Concentration Inequalities and Empirical Processes Theory Applied to the Analysis of Learning Algorithms*. PhD thesis, Ecole Polytechnique, 2002.

N. Cesa-Bianchi and G. Lugosi. *Prediction, learning, and games*. Cambridge University Press, 2006.

N. Cristianini and J. Shawe-Taylor. *Kernel Methods for Pattern Analysis*. Cambridge University Press, 2004.

Y. Freund and R. E. Schapire. Large margin classification using the perceptron algorithm. *Machine Learning*, 37(3):277–296, 1999.

Y. Freund and R.E. Schapire. A decision-theoretic generalization of on-line learning and an application to boosting. *Journal of Computer and System Sciences*, 55(1):119–139, August 1997.

C. Gentile. The robustness of the p-norm algorithms. *Machine Learning*, 53(3):265–299, 2003.

A. Kalai, A.R. Klivans, Y. Mansour, and R. Servedio. Agnostically learning halfspaces. In *Proceedings of the 46th Foundations of Computer Science (FOCS)*, 2005.

A.T. Kalai and R. Sastry. The isotron algorithm: High-dimensional isotonic regression. In *Proceedings of the 22th Annual Conference on Learning Theory*, 2009.

A.R. Klivans, P.M. Long, and R.A. Servedio. Learning halfspaces with malicious noise. *The Journal of Machine Learning Research*, 10:2715–2740, 2009.

P.M. Long and R.A. Servedio. Learning large-margin halfspaces with more malicious noise. In *NIPS*, 2011.

F. Rosenblatt. The perceptron: A probabilistic model for information storage and organization in the brain. *Psychological Review*, 65:386–407, 1958. (Reprinted in *Neurocomputing* (MIT Press, 1988).).

B. Schölkopf and A. J. Smola. *Learning with Kernels: Support Vector Machines, Regularization, Optimization and Beyond*. MIT Press, 2002.

R.A. Servedio. Smooth boosting and learning with malicious noise. *Journal of Machine Learning Research*, 4: 633–648, 2003.

S. Shalev-Shwartz, O. Shamir, and K. Sridharan. Learning kernel-based halfspaces with the 0-1 loss. *SIAM Journal on Computing*, 40:1623–1646, 2011.

L. G. Valiant. Learning disjunctions of conjunctions. In *Proceedings of the 9th International Joint Conference on Artificial Intelligence*, pages 560–566, August 1985.

V. N. Vapnik. *Statistical Learning Theory*. Wiley, 1998.

T. Zhang. Statistical behavior and consistency of classification methods based on convex risk minimization. *The Annals of Statistics*, 32:56–85, 2004.

